# Mixability in Statistical Learning

**Tim van Erven**
Université Paris-Sud, France
tim@timvanerven.nl

**Peter D. Grünwald**
CWI and Leiden University, the Netherlands
pdg@cwi.nl

**Mark D. Reid**
ANU and NICTA, Australia
Mark.Reid@anu.edu.au

**Robert C. Williamson**
ANU and NICTA, Australia
Bob.Williamson@anu.edu.au

## Abstract

Statistical learning and sequential prediction are two different but related formalisms to study the quality of predictions. Mapping out their relations and transferring ideas is an active area of investigation. We provide another piece of the puzzle by showing that an important concept in sequential prediction, the mixability of a loss, has a natural counterpart in the statistical setting, which we call *stochastic mixability*. Just as ordinary mixability characterizes fast rates for the worst-case regret in sequential prediction, stochastic mixability characterizes fast rates in statistical learning. We show that, in the special case of log-loss, stochastic mixability reduces to a well-known (but usually unnamed) martingale condition, which is used in existing convergence theorems for minimum description length and Bayesian inference. In the case of $0/1$-loss, it reduces to the margin condition of Mammen and Tsybakov, and in the case that the model under consideration contains all possible predictors, it is equivalent to ordinary mixability.

## 1  Introduction

In *statistical learning* (also called batch learning) [1] one obtains a random sample $(X_1, Y_1), \ldots, (X_n, Y_n)$ of independent pairs of observations, which are all distributed according to the same distribution $P^*$. The goal is to select a function $\hat{f}$ that maps $X$ to a prediction $\hat{f}(X)$ of $Y$ for a new pair $(X, Y)$ from the same $P^*$. The quality of $\hat{f}$ is measured by its *excess risk*, which is the expectation of its *loss* $\ell(Y, \hat{f}(X))$ minus the expected loss of the best prediction function $f^*$ in a given class of functions $\mathcal{F}$. Analysis in this setting usually involves giving guarantees about the performance of $\hat{f}$ in the worst case over the choice of the distribution of the data.

In contrast, the setting of *sequential prediction* (also called online learning) [2] makes no probabilistic assumptions about the source of the data. Instead, pairs of observations $(x_t, y_t)$ are assumed to become available one at a time, in rounds $t = 1, \ldots, n$, and the goal is to select a function $\hat{f}_t$ just before round $t$, which maps $x_t$ to a prediction of $y_t$. The quality of predictions $\hat{f}_1, \ldots, \hat{f}_n$ is evaluated by their *regret*, which is the sum of their losses $\ell(y_1, \hat{f}_1(x_1)), \ldots, \ell(y_n, \hat{f}_n(x_n))$ on the actual observations minus the total loss of the best fixed prediction function $f^*$ in a class of functions $\mathcal{F}$. In sequential prediction the usual analysis involves giving guarantees about the performance of $\hat{f}_1, \ldots, \hat{f}_n$ in the worst case over all possible realisations of the data. When stating rates of convergence, we will divide the worst-case regret by $n$, which makes the rates comparable to rates in the statistical learning setting.

Mapping out the relations between statistical learning and sequential prediction is an active area of investigation, and several connections are known. For example, using any of a variety of *online-*

*to-batch conversion* techniques [3], any sequential predictions $\hat{f}_1, \ldots, \hat{f}_n$ may be converted into a single statistical prediction $\hat{f}$ and the statistical performance of $\hat{f}$ is bounded by the sequential prediction performance of $\hat{f}_1, \ldots, \hat{f}_n$. Moreover, a deep understanding of the relation between worst-case rates in both settings is provided by Abernethy, Agarwal, Bartlett and Rakhlin [4]. Amongst others, their results imply that for many loss functions the worst-case rate in sequential prediction exceeds the worst-case rate in statistical learning.

**Fast Rates** In sequential prediction with a finite class $\mathcal{F}$, it is known that the worst-case regret can be bounded by a constant if and only if the loss $\ell$ has the property of being *mixable* [5, 6] (subject to mild regularity conditions on the loss). Dividing by $n$, this corresponds to $O(1/n)$ rates, which is fast compared to the usual $O(1/\sqrt{n})$ rates.

In statistical learning, there are two kinds of conditions that are associated with fast rates. First, for $0/1$-loss, fast rates (faster than $O(1/\sqrt{n})$) are associated with Mammen and Tsybakov's *margin condition* [7, 8], which depends on a parameter $\kappa$. In the nicest case, $\kappa = 1$ and then $O(1/n)$ rates are possible. Second, for log(arithmic) loss there is a single supermartingale condition that is essential to obtain fast rates in all convergence proofs of two-part minimum description length (MDL) estimators, and in many convergence proofs of Bayesian estimators. This condition, used by e.g. [9, 10, 11, 12, 13, 14], sometimes remains implicit (see Example 1 below) and usually goes unnamed. A special case has been called the 'supermartingale property' by Chernov, Kalnishkan, Zhdanov and Vovk [15]. Audibert [16] also introduced a closely related condition, which does seem subtly different however.

**Our Contribution** We define the notion of *stochastic mixability* of a loss $\ell$, set of predictors $\mathcal{F}$, and distribution $P^*$, which we argue to be the natural analogue of mixability for the statistical setting on two grounds: first, we show that it is closely related to both the supermartingale condition and the margin condition, the two properties that are known to be related to fast rates; second, we show that it shares various essential properties with ordinary mixability and in specific cases is even equivalent to ordinary mixability.

To support the first part of our argument, we show the following: **(a)** for bounded losses (including $0/1$-loss), stochastic mixability is equivalent to the best case ($\kappa = 1$) of a generalization of the margin condition; other values of $\kappa$ may be interpreted in terms of a slightly relaxed version of stochastic mixability; **(b)** for log-loss, stochastic mixability reduces to the supermartingale condition; **(c)** in general, stochastic mixability allows uniform $O(\log |\mathcal{F}_n|/n)$-statistical learning rates to be achieved, where $|\mathcal{F}_n|$ is the size of a sub-model $\mathcal{F}_n \subset \mathcal{F}$ considered at sample size $n$. Finally, **(d)** if stochastic mixability does not hold, then in general $O(\log |\mathcal{F}_n|/n)$-statistical learning rates cannot be achieved, at least not for $0/1$-loss or for log-loss.

To support the second part of our argument, we show: **(e)** if the set $\mathcal{F}$ is 'full', i.e. it contains all prediction functions for the given loss, then stochastic mixability turns out to be formally equivalent to ordinary mixability (if $\mathcal{F}$ is not full, then either condition may hold without the other). We choose to call our property stochastic mixability rather than, say, 'generalized margin condition for $\kappa = 1$' or 'generalized supermartingale condition', because **(f)** we also show that the general condition can be formulated in an alternative way (Theorem 2) that directly indicates a strong relation to ordinary mixability, and **(g)** just like ordinary mixability, it can be interpreted as the requirement that a set of so-called pseudo-likelihoods is (effectively) convex.

We note that special cases of results (a)–(e) already follow from existing work of many other authors; we provide a detailed comparison in Section 7. Our contributions are to generalize these results, and to relate them to each other, to the notion of mixability from sequential prediction, and to the interpretation in terms of convexity of a set of pseudo-likelihoods. This leads to our central conclusion: the concept of stochastic mixability is closely related to mixability and plays a fundamental role in achieving fast rates in the statistical learning setting.

**Outline** In §2 we define both ordinary mixability and stochastic mixability. We show that two of the standard ways to express mixability have natural analogues that express stochastic mixability (leading to (f)). In example 1 we specialize the definition to log-loss and explain its importance in the literature on MDL and Bayesian inference, leading to (b). A third interpretation of mixability and standard mixability in terms of sets (g) is described in §3. The equivalence between mixability

and stochastic mixability if $\mathcal{F}$ is full is presented in §4 where we also show that the equivalence need not hold if $\mathcal{F}$ is not full (e). In §5, we turn our attention to a version of the margin condition that does not assume that $\mathcal{F}$ contains the Bayes optimal predictor and we show that (a slightly relaxed version of) stochastic mixability is equivalent to the margin condition, taking care of (a). We show (§6) that if stochastic mixability holds, $O(\log|\mathcal{F}_n|/n)$-rates can always be achieved (c), and that in some cases in which it does not hold, $O(\log|\mathcal{F}_n|/n)$-rates cannot be achieved (d). Finally (§7) we connect our results to previous work in the literature. Proofs omitted from the main body of the paper are in the supplementary material.

## 2 Mixability and Stochastic Mixability

We now introduce the notions of mixability and stochastic mixability, showing two equivalent formulations of the latter.

### 2.1 Mixability

A *loss function* $\ell \colon \mathcal{Y} \times \mathcal{A} \to [0, \infty]$ is a nonnegative function that measures the quality of a prediction $a \in \mathcal{A}$ when the true outcome is $y \in \mathcal{Y}$ by $\ell(y, a)$. We will assume that all spaces come equipped with appropriate $\sigma$-algebras, so we may define distributions on them, and that the loss function $\ell$ is measurable.

**Definition 1** (Mixability). For $\eta > 0$, a loss $\ell$ is called *$\eta$-mixable* if for any distribution $\pi$ on $\mathcal{A}$ there exists a single prediction $a_\pi$ such that

$$\ell(y, a_\pi) \leq -\frac{1}{\eta} \ln \int e^{-\eta \ell(y,a)} \pi(\mathrm{d}a) \qquad \text{for all } y. \tag{1}$$

It is called *mixable* if there exists an $\eta > 0$ such that it is $\eta$-mixable.

Let $A$ be a random variable with distribution $\pi$. Then (1) may be rewritten as

$$\mathbf{E}_\pi \left[ \frac{e^{-\eta \ell(y,A)}}{e^{-\eta \ell(y,a_\pi)}} \right] \leq 1 \qquad \text{for all } y. \tag{2}$$

### 2.2 Stochastic Mixability

Let $\mathcal{F}$ be a set of *predictors* $f \colon \mathcal{X} \to \mathcal{A}$, which are measurable functions that map any input $x \in \mathcal{X}$ to a prediction $f(x)$. For example, if $\mathcal{A} = \mathcal{Y} = \{0, 1\}$ and the loss is the 0/1-loss, $\ell_{0/1}(y, a) = \mathbf{1}\{y \neq a\}$, then the predictors are classifiers. Let $P^*$ be the distribution of a pair of random variables $(X, Y)$ with values in $\mathcal{X} \times \mathcal{Y}$. Most expectations in the paper are with respect to $P^*$. Whenever this is not the case we will add a subscript to the expectation operator, as in (2).

**Definition 2** (Stochastic Mixability). For any $\eta \geq 0$, we say that $(\ell, \mathcal{F}, P^*)$ is *$\eta$-stochastically mixable* if there exists an $f^* \in \mathcal{F}$ such that

$$\mathbf{E} \left[ \frac{e^{-\eta \ell(Y,f(X))}}{e^{-\eta \ell(Y,f^*(X))}} \right] \leq 1 \qquad \text{for all } f \in \mathcal{F}. \tag{3}$$

We call $(\ell, \mathcal{F}, P^*)$ *stochastically mixable* if there exists an $\eta > 0$ such that it is $\eta$-stochastically mixable.

By Jensen's inequality, we see that (3) implies $1 \geq \mathbf{E} \left[ \frac{e^{-\eta \ell(Y,f(X))}}{e^{-\eta \ell(Y,f^*(X))}} \right] \geq e^{\mathbf{E}[\eta(\ell(Y,f^*(X)) - \ell(Y,f(X)))]}$, so that

$$\mathbf{E}[\ell(Y, f^*(X))] \leq \mathbf{E}[\ell(Y, f(X)))] \qquad \text{for all } f \in \mathcal{F},$$

and hence the definition of stochastic mixability presumes that $f^*$ minimizes $\mathbf{E}[\ell(Y, f(X))]$ over all $f \in \mathcal{F}$. We will assume throughout the paper that such an $f^*$ exists, and that $\mathbf{E}[\ell(Y, f^*(X))] < \infty$.

The larger $\eta$, the stronger the requirement of $\eta$-stochastic mixability:

**Proposition 1.** *Any triple $(\ell, \mathcal{F}, P^*)$ is 0-stochastically mixable. And if $0 < \gamma < \eta$, then $\eta$-stochastic mixability implies $\gamma$-stochastic mixability.*

**Example 1** (Log-loss). Let $\mathcal{F}$ be a set of conditional probability densities and let $\ell_{\log}$ be log-loss, i.e. $\mathcal{A}$ is the set of densities on $\mathcal{Y}$, $f(x)(y)$ is written, as usual, as $f(y \mid x)$, and $\ell_{\log}(y, f(x)) := -\ln f(y \mid x)$. For log-loss, statistical learning becomes equivalent to conditional density estimation with random design (see, e.g., [14]). Equation 3 now becomes equivalent to

$$A_\eta(f^* \| f) := \mathbf{E} \left( \frac{f(Y \mid X)}{f^*(Y \mid X)} \right)^\eta \leq 1. \tag{4}$$

$A_\eta$ has been called the *generalized Hellinger affinity* [12] in the literature. If the model is correct, i.e. it contains the true conditional density $p^*(y \mid x)$, then, because the log-loss is a proper loss [17] we must have $f^* = p^*$ and then, for $\eta = 1$, trivially $A_\eta(f \| f^*) = 1$ for all $f \in \mathcal{F}$. Thus if the model $\mathcal{F}$ is correct, then the log-loss is $\eta$-stochastically mixable for $\eta = 1$. In that case, for $\eta = 1/2$, $A_\eta$ turns into the standard definition of Hellinger affinity [10].

Equation 4 — which just expresses 1-stochastic mixability for log-loss — is used in all previous convergence theorems for 2-part MDL density estimation [10, 12, 11, 18], and, more implicitly, in various convergence theorems for Bayesian procedures, including the pioneering paper by Doob [9]. All these results assume that the model $\mathcal{F}$ is correct, but, if one studies the proofs, one finds that the assumption is only needed to establish that (4) holds for $\eta = 1$. For example, as first noted by [12], if $\mathcal{F}$ is a convex set of densities, then (4) also holds for $\eta = 1$, even if the model is incorrect, and, indeed, two-part MDL converges at fast rates in such cases (see [14] for a precise definition of what this means, as well as more general treatment of (4)). Kleijn and Van der Vaart [13], in their extensive analysis of Bayesian nonparametric inference if the model is wrong, also use the fact that (4) holds with $\eta = 1$ for convex models to show that fast posterior concentration rates hold for such models even if they do not contain the true $p^*$.

The definition of stochastic mixability looks similar to (2), but whereas $\pi$ is a distribution on predictions, $P^*$ is a distribution on outcomes $(X, Y)$. Thus at first sight the resemblance appears to be only superficial. It is therefore quite surprising that stochastic mixability can also be expressed in a way that looks like (1), which provides a first hint that the relation goes deeper.

**Theorem 2.** *Let $\eta > 0$. Then $(\ell, \mathcal{F}, P^*)$ is $\eta$-stochastically mixable if and only if for any distribution $\pi$ on $\mathcal{F}$ there exists a single predictor $f^* \in \mathcal{F}$ such that*

$$\mathbf{E}\left[\ell(Y, f^*(X))\right] \leq \mathbf{E}\left[-\frac{1}{\eta}\ln \int e^{-\eta\ell(Y, f(X))} \pi(\mathrm{d}f)\right]. \tag{5}$$

Notice that, without loss of generality, we can always choose $f^*$ to be the minimizer of $\mathbf{E}[\ell(Y, f(X))]$. Then $f^*$ does not depend on $\pi$.

## 3 The Convexity Interpretation

There is a third way to express mixability, as the convexity of a set of so-called pseudo-likelihoods. We will now show that stochastic mixability can also be interpreted as convexity of the corresponding set in the statistical learning setting.

Following Chernov *et al.* [15], we first note that the essential feature of a loss $\ell$ with corresponding set of predictions $\mathcal{A}$ is the set of *achievable losses* they induce:

$$\mathcal{L} = \{l \colon \mathcal{Y} \to [0, \infty] \mid \exists a \in \mathcal{A} \colon l(y) = \ell(y, a) \text{ for all } y \in \mathcal{Y}\}.$$

If we would reparametrize the loss by a different set of predictions $\mathcal{A}'$, while keeping $\mathcal{L}$ the same, then essentially nothing would change. For example, for $0/1$-loss standard ways to parametrize predictions are by $\mathcal{A} = \{0, 1\}$, by $\mathcal{A} = \{-1, +1\}$ or by $\mathcal{A} = \mathbb{R}$ with the interpretation that predicting $a \geq 0$ maps to the prediction 1 and $a < 0$ maps to the prediction 0. Of course these are all equivalent, because $\mathcal{L}$ is the same.

It will be convenient to consider the set of functions that lie above the achievable losses in $\mathcal{L}$:

$$\mathcal{S} = \mathcal{S}_\ell = \{l \colon \mathcal{Y} \to [0, \infty] \mid \exists l' \in \mathcal{L} \colon l(y) \geq l'(y) \text{ for all } y \in \mathcal{Y}\},$$

Chernov *et al.* call this the *super prediction set*. It plays a role similar to the role of the epigraph of a function in convex analysis. Let $\eta > 0$. Then with each element $l \in \mathcal{S}$ in the super prediction

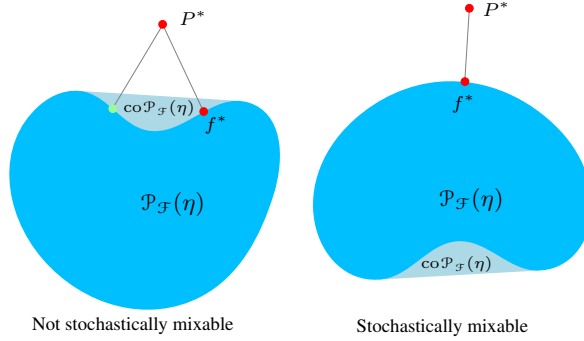

Figure 1: The relation between convexity and stochastic mixability for log-loss, $\eta = 1$ and $\mathcal{X} = \{x\}$ a singleton, in which case $P^*$ and the elements of $\mathcal{P}_{\mathcal{F}}(\eta)$ can all be interpreted as distributions on $\mathcal{Y}$.

set, we associate a *pseudo-likelihood* $p(y) = e^{-\eta l(y)}$. Note that $0 \le p(y) \le 1$, but it is generally not the case that $\int p(y) \mu(\mathrm{d}y) = 1$ for some reference measure $\mu$ on $\mathcal{Y}$, so $p(y)$ is not normalized. Let $e^{-\eta \mathcal{S}} = \{e^{-\eta l} \mid l \in \mathcal{S}\}$ denote the set of all such pseudo-likelihoods. By multiplying (1) by $-\eta$ and exponentiating, it can be shown that $\eta$-*mixability is exactly equivalent to the requirement that* $e^{-\eta \mathcal{S}}$ *is convex* [2, 15]. And like for the first two expressions of mixability, there is an analogous convexity interpretation for stochastic mixability.

In order to define pseudo-likelihoods in the statistical setting, we need to take into account that the predictions $f(X)$ of the predictors in $\mathcal{F}$ are not deterministic, but depend on $X$. Hence we define *conditional pseudo-likelihoods* $p(Y|X) = e^{-\eta \ell(Y, f(X))}$. (See also Example 1.) There is no need to introduce a conditional analogue of the super prediction set. Instead, let $\mathcal{P}_{\mathcal{F}}(\eta) = \{e^{-\eta \ell(Y, f(X))} \mid f \in \mathcal{F}\}$ denote the set of all conditional pseudo-likelihoods. For $\lambda \in [0, 1]$, a convex combination of any two $p_0, p_1 \in \mathcal{P}_{\mathcal{F}}(\eta)$ can be defined as $p_\lambda(Y|X) = (1 - \lambda)p_0(Y|X) + \lambda p_1(Y|X)$. And consequently, we may speak of the convex hull $\mathrm{co}\,\mathcal{P}_{\mathcal{F}}(\eta) = \{p_\lambda \mid p_0, p_1 \in \mathcal{P}_{\mathcal{F}}(\eta), \lambda \in [0, 1]\}$ of $\mathcal{P}_{\mathcal{F}}(\eta)$.

**Corollary 3.** *Let $\eta > 0$. Then $\eta$-stochastic mixability of $(\ell, \mathcal{F}, P^*)$ is equivalent to the requirement that*

$$\min_{p \in \mathcal{P}_{\mathcal{F}}(\eta)} \mathbf{E}\left[\tfrac{-1}{\eta} \ln p(Y|X)\right] = \min_{p \in \mathrm{co}\,\mathcal{P}_{\mathcal{F}}(\eta)} \mathbf{E}\left[\tfrac{-1}{\eta} \ln p(Y|X)\right]. \tag{6}$$

*Proof.* This follows directly from Theorem 2 after rewriting it in terms of conditional pseudo-likelihoods. $\qquad\square$

Notice that the left-hand side of (6) equals $\mathbf{E}[\ell(Y, f^*(X))]$, which does not depend on $\eta$.

Equation 6 expresses that the convex hull operator has no effect, which means that $\mathcal{P}_{\mathcal{F}}(\eta)$ looks convex from the perspective of $P^*$. See Figure 1 for an illustration for log-loss. Thus we obtain an interpretation of $\eta$-stochastic mixability as effective convexity of the set of pseudo-likelihoods $\mathcal{P}_{\mathcal{F}}(\eta)$ with respect to $P^*$.

Figure 1 suggests that $f^*$ should be unique if the loss is stochastically mixable, which is almost right. It is in fact the loss $\ell(Y, f^*(X))$ of $f^*$ that is unique (almost surely):

**Corollary 4.** *If $(\ell, \mathcal{F}, P^*)$ is stochastically mixable and there exist $f^*, g^* \in \mathcal{F}$ such that $\mathbf{E}[\ell(Y, f^*(X))] = \mathbf{E}[\ell(Y, g^*(X))] = \min_{f \in \mathcal{F}} \mathbf{E}[\ell(Y, f(X))]$, then $\ell(Y, f^*(X)) = \ell(Y, g^*(X))$ almost surely.*

*Proof.* Let $\pi(f^*) = \pi(g^*) = 1/2$. Then, by Theorem 2 and (strict) convexity of $-\ln$,

$$\min_{f \in \mathcal{F}} \mathbf{E}[\ell(Y, f(X))] \le \mathbf{E}\left[-\frac{1}{\eta} \ln \left(\frac{1}{2} e^{-\eta \ell(Y, f^*(X))} + \frac{1}{2} e^{-\eta \ell(Y, g^*(X))}\right)\right]$$

$$\le \mathbf{E}\left[\frac{1}{2}\ell(Y, f^*(X)) + \frac{1}{2}\ell(Y, g^*(X))\right] = \min_{f \in \mathcal{F}} \mathbf{E}[\ell(Y, f(X))].$$

Hence both inequalities must hold with equality. For the second inequality this is only the case if $\ell(Y, f^*(X)) = \ell(Y, g^*(X))$ almost surely, which was to be shown. $\qquad\square$

## 4   When Mixability and Stochastic Mixability Are the Same

Having observed that mixability and stochastic mixability of a loss share several common features, we now show that in specific cases the two concepts even coincide. More specifically, Theorem 5 below shows that a loss $\ell$ (meeting two requirements) is $\eta$-mixable if and only if it is $\eta$-stochastically mixable relative to $\mathcal{F}_{\mathrm{full}}$, the set of all functions from $\mathcal{X}$ to $\mathcal{A}$, and all distributions $P^*$. To avoid measurability issues, we will assume that $\mathcal{X}$ is countable throughout this section.

The two conditions we assume of $\ell$ are both related to its set of pseudo-likelihoods $\Phi := e^{-\eta \mathcal{S}}$, which was defined in Section 3. The first condition is that $\Phi$ is *closed*. When $\mathcal{Y}$ is infinite, we mean closed relative to the topology for the supremum norm $\|p\|_\infty = \sup_{y \in \mathcal{Y}} |p(y)|$. The second, more technical condition is that $\Phi$ is *pre-supportable*. That is, for every pseudo-likelihood $p \in \Phi$, its pre-image $s \in \mathcal{S}$ (defined for each $y \in \mathcal{Y}$ by $s(y) := -\frac{1}{\eta} \ln p(y)$) is supportable. Here, a point $s \in \mathcal{S}$ is *supportable* if it is optimal for some distribution $P_Y^*$ over $\mathcal{Y}$ – that is, if there exists a distribution $P_Y^*$ over $\mathcal{Y}$ such that $\mathbf{E}_{P_Y^*}[s(Y)] \leq \mathbf{E}_{P_Y^*}[t(Y)]$ for all $t \in \mathcal{S}$. This is the case, for example, for all proper losses [17].

We say $(\ell, \mathcal{F})$ is $\eta$-stochastically mixable if $(\ell, \mathcal{F}, P^*)$ is $\eta$-stochastically mixable for all distributions $P^*$ on $\mathcal{X} \times \mathcal{Y}$.

**Theorem 5.** *Suppose $\mathcal{X}$ is countable. Let $\eta > 0$ and suppose $\ell$ is a loss such that its pseudo-likelihood set $e^{-\eta \mathcal{S}}$ is closed and pre-supportable. Then $(\ell, \mathcal{F}_{\mathrm{full}})$ is $\eta$-stochastically mixable if and only if $\ell$ is $\eta$-mixable.*

This result generalizes Theorem 9 and Lemma 11 by Chernov *et al.* [15] from finite $\mathcal{Y}$ to arbitrary continuous $\mathcal{Y}$, which they raised as an open question. In their setting, there are no explanatory variables $x$, which may be emulated in our framework by letting $\mathcal{X}$ contain only a single element. Their conditions also imply (by their Lemma 10) that the loss $\ell$ is proper, which implies that $e^{-\eta \mathcal{S}}$ is closed and pre-supportable. We note that for proper losses $\eta$-mixability is especially well understood [19].

The proof of Theorem 5 is broken into two lemmas (the proofs of which are in the supplementary material). The first establishes conditions for when mixability implies stochastic mixability, borrowing from a similar result for log-loss by Li [12].

**Lemma 6.** *Let $\eta > 0$. Suppose the Bayes optimal predictor $f_B^*(x) \in \arg\min_{a \in \mathcal{A}} \mathbf{E}[\ell(Y, a)|X = x]$ is in the model: $f_B^* = f^* \in \mathcal{F}$. If $\ell$ is $\eta$-mixable, then $(\ell, \mathcal{F}, P^*)$ is $\eta$-stochastically mixable.*

The second lemma shows that stochastic mixability implies mixability.

**Lemma 7.** *Suppose the conditions of Theorem 5 are satisfied. If $(\ell, \mathcal{F}_{\mathrm{full}})$ is $\eta$-stochastically mixable, then it is $\eta$-mixable.*

The above two lemmata are sufficient to prove the equivalence of stochastic and ordinary mixability.

*Proof of Theorem 5.* In order to show that $\eta$-mixability of $\ell$ implies $\eta$-stochastic mixability of $(\ell, \mathcal{F}_{\mathrm{full}})$ we note that the Bayes-optimal predictor $f_B^*$ for any $\ell$ and $P^*$ must be in $\mathcal{F}_{\mathrm{full}}$ and so Lemma 6 implies $(\ell, \mathcal{F}_{\mathrm{full}}, P^*)$ is $\eta$-stochastically mixable for any distribution $P^*$. Conversely, that $\eta$-stochastic mixability of $(\ell, \mathcal{F}_{\mathrm{full}})$ implies the $\eta$-mixability of $\ell$ follows immediately from Lemma 7. $\qquad\square$

**Example 2** (if $\mathcal{F}$ is not full)**.** In this case, we can have either stochastic mixability without ordinary mixability or the converse. Consider a loss function $\ell$ that is not mixable in the ordinary sense, e.g. $\ell = \ell_{0/1}$, the 0/1-loss [6], and a set $\mathcal{F}$ consisting of just a single predictor. Then clearly $\ell$ is stochastically mixable relative to $\mathcal{F}$. This is, of course, a trivial case. We do not know whether we can have stochastic mixability without ordinary mixability in nontrivial cases, and plan to investigate this for future work. For the converse, we know that it does hold in nontrivial cases: consider the log-loss $\ell_{\log}$ which is 1-mixable in the standard sense (Example 1). Let $\mathcal{Y} = \{0, 1\}$ and let the model $\mathcal{F}$ be a set of conditional probability mass functions $\{f_\theta \mid \theta \in \Theta\}$ where $\Theta$ is the

set of all classifiers, i.e. all functions $\mathcal{X} \to \mathcal{Y}$, and $f_\theta(y \mid x) := e^{-\ell_{0/1}(y,\theta(x))}/(1 + e^{-1})$ where $\ell_{0/1}(y, \hat{y}) = \mathbf{1}\{y \neq \hat{y}\}$ is the 0/1-loss. Then log-loss becomes an affine function of 0/1-loss: for each $\theta \in \Theta$, $\ell_{\log}(Y, f_\theta(X)) = \ell_{0/1}(Y, \theta(X)) + C$ with $C = \ln(1 + e^{-1})$ [14]. Because 0/1-loss is not standard mixable, by Theorem 5, 0/1-loss is not stochastically mixable relative to $\Theta$. But then we must also have that log-loss is not stochastically mixable relative to $\mathcal{F}$.

# 5   Stochastic Mixability and the Margin Condition

The *excess risk* of any $f$ compared to $f^*$ is the mean of the *excess loss* $\ell(Y, f(X)) - \ell(Y, f^*(X))$:

$$d(f, f^*) = \mathbf{E}\left[\ell(Y, f(X)) - \ell(Y, f^*(X))\right].$$

We also define the expected square of the excess loss, which is closely related to its variance:

$$V(f, f^*) = \mathbf{E}\left(\ell(Y, f(X)) - \ell(Y, f^*(X))\right)^2.$$

Note that, for 0/1-loss, $V(f, f^*) = P^*(f(X) \neq f^*(X))$ is the probability that $f$ and $f^*$ disagree.

The *margin condition*, introduced by Mammen and Tsybakov [7, 8] for 0/1-loss, is satisfied with constants $\kappa \geq 1$ and $c_0 > 0$ if

$$c_0 V(f, f^*)^\kappa \leq d(f, f^*) \qquad \text{for all } f \in \mathcal{F}. \tag{7}$$

Unlike Mammen and Tsybakov, we do not assume that $\mathcal{F}$ necessarily contains the Bayes predictor, which minimizes the risk over all possible predictors. The same generalization has been used in the context of model selection by Arlot and Bartlett [20].

*Remark* 1. In some practical cases, the margin condition only holds for a subset of the model such that $V(f, f^*) \leq \epsilon_0$ for some $\epsilon_0 > 0$ [8]. In such cases, the discussion below applies to the same subset.

Stochastic mixability, as we have defined it, is directly related to the margin condition for the case $\kappa = 1$. In order to relate it to other values of $\kappa$, we need a little more flexibility: for given $\epsilon \geq 0$ and $(\ell, \mathcal{F}, P^*)$, we define

$$\mathcal{F}_\epsilon = \{f^*\} \cup \{f \in \mathcal{F} \mid d(f, f^*) \geq \epsilon\}, \tag{8}$$

which excludes a band of predictors that approximate the best predictor in the model to within excess risk $\epsilon$.

**Theorem 8.** *Suppose a loss $\ell$ takes values in $[0, V]$ for $0 < V < \infty$. Fix a model $\mathcal{F}$ and distribution $P^*$. Then the margin condition (7) is satisfied if and only if there exists a constant $C > 0$ such that, for all $\epsilon > 0$, $(\ell, \mathcal{F}_\epsilon, P^*)$ is $\eta$-stochastically mixable for $\eta = C\epsilon^{(\kappa-1)/\kappa}$. In particular, if the margin condition is satisfied with constants $\kappa$ and $c_0$, we can take $C = \min\left\{\frac{V^2 c_0^{1/\kappa}}{e^V - V - 1}, \frac{1}{V^{(\kappa-1)/\kappa}}\right\}$.*

This theorem gives a new interpretation of the margin condition as the rate at which $\eta$ has to go to 0 when the model $\mathcal{F}$ is approximated by $\eta$-stochastically mixable models $\mathcal{F}_\epsilon$. By the following corollary, proved in the additional material, stochastic mixability of the whole model $\mathcal{F}$ is equivalent to the best case of the margin condition.

**Corollary 9.** *Suppose $\ell$ takes values in $[0, V]$ for $0 < V < \infty$. Then $(\ell, \mathcal{F}, P^*)$ is stochastically mixable if and only if there exists a constant $c_0 > 0$ such that the margin condition (7) is satisfied with $\kappa = 1$.*

# 6   Connection to Uniform $O(\log |\mathcal{F}_n|/n)$ Rates

Let $\ell$ be a bounded loss function. Assume that, at sample size $n$, an estimator $\hat{f}$ (statistical learning algorithm) is used based on a finite model $\mathcal{F}_n$, where we allow the size $|\mathcal{F}_n|$ to grow with $n$. Let, for all $n$, $\mathcal{P}_n$ be any set of distributions on $\mathcal{X} \times \mathcal{Y}$ such that for all $P^* \in \mathcal{P}_n$, the generalized margin condition (7) holds for $\kappa = 1$ and uniform constant $c_0$ not depending on $n$, with model $\mathcal{F}_n$. In the case of 0/1-loss, the results of e.g. Tsybakov [8] suggest that there exist estimators

$\hat{f}_n : (\mathcal{X} \times \mathcal{Y})^n \to \mathcal{F}_n$ that achieve a convergence rate of $O(\log |\mathcal{F}_n|/n)$, uniformly for all $P^* \in \mathcal{P}$; that is,

$$\sup_{P^* \in \mathcal{P}_n} \mathbf{E}_{P^*}[d(\hat{f}_n, f^*)] = O(\log |\mathcal{F}_n|/n). \qquad (9)$$

This can indeed be proven, for general loss functions, using Theorem 4.2. of Zhang [21] and with $\hat{f}_n$ set to Zhang's information-risk-minimization estimator (to see this, at sample size $n$ apply Zhang's result with $\alpha$ set to 0 and a prior $\pi$ that is uniform on $\mathcal{F}_n$, so that $-\log \pi(f) = \log |\mathcal{F}_n|$ for any $f \in \mathcal{F}_n$). By Theorem 8, this means that, for any bounded loss function $\ell$, if, for some $\eta > 0$, all $n$, we have that $(\ell, \mathcal{F}_n, P^*)$ is $\eta$-stochastically mixable for all $P^* \in \mathcal{P}_n$, then Zhang's estimator satisfies (9). Hence, for bounded loss functions, stochastic mixability implies a uniform $O(\log |\mathcal{F}_n|/n)$ rate.

A connection between stochastic mixability and fast rates is also made by Grünwald [14], who introduces some slack in the definition (allowing the number 1 in (3) to be slightly larger) and uses the convexity interpretation from Section 3 to empirically determine the largest possible value for $\eta$. His Theorem 2, applied with a slack set to 0, implies an in-probability version of Zhang's result above.

**Example 3.** We just explained that, if $\ell$ is stochastically mixable relative to $\mathcal{F}_n$, then uniform $O(\log |\mathcal{F}_n|/n)$ rates can be achieved. We now illustrate that if this is not the case, then, at least if $\ell$ is 0/1-loss or if $\ell$ is log-loss, uniform $O(\log |\mathcal{F}_n|/n)$ rates cannot be achieved in general. To see this, let $\Theta_n$ be a finite set of classifiers $\theta : \mathcal{X} \to \mathcal{Y}$, $\mathcal{Y} = \{0, 1\}$ and let $\ell$ be 0/1-loss. Let for each $n$, $\hat{f}_n : (\mathcal{X} \times \mathcal{Y})^n \to \mathcal{F}_n$ be some arbitrary estimator. It is known from e.g. the work of Vapnik [22] that for every sequence of estimators $\hat{f}_1, \hat{f}_2, \ldots$, there exist a sequence $\Theta_1, \Theta_2, \ldots$, all finite, and a sequence $P_1^*, P_2^*, \ldots$ such that

$$\frac{\mathbf{E}_{P_n^*}[d(\hat{f}_n, f^*)]}{\log |\Theta_n|/n} \to \infty.$$

Clearly then, by Zhang's result above, there cannot be an $\eta$ such that for all $n$, $(\ell, \Theta_n, P_n^*)$ is $\eta$-stochastically mixable. This establishes that if stochastic mixability does not hold, then uniform rates of $O(\log |\mathcal{F}_n|/n)$ are not achievable in general for 0/1-loss. By the construction of Example 2, we can modify $\Theta_n$ into a set of corresponding log-loss predictors $\mathcal{F}_n$ so that the log-loss $\ell_{\log}$ becomes an affine function of the 0/1-loss; thus, there still is no $\eta$ such that for all $n$, $(\ell_{\log}, \mathcal{F}_n, P_n^*)$ is $\eta$-mixable, and the sequence of estimators still cannot achieve uniform a $O(\log |\mathcal{F}_n|/n)$ rate with log-loss either.

# 7   Discussion — Related Work

Let us now return to the summary of our contributions which we provided as items (a)—(g) in §1. We note that slight variations of our formula (3) for stochastic mixability already appear in [14] (but there no connections to ordinary mixability are made) and [15] (but there it is just a tool for the worst-case sequential setting, and no connections to fast rates in statistical learning are made). Equation 3 looks completely different from the margin condition, yet results connecting the two, somewhat similar to (a), albeit very implicitly, already appear in [23] and [24]. Also, the paper by Grünwald [14] contains a connection between the margin condition somewhat similar to Theorem 8, but involving a significantly weaker version of stochastic mixability in which the inequality (3) only holds with some slack. Result (b) is trivial given Definition 2; (c) is a consequence of Theorem 4.2. of [21] when combined with (a) (see Section 6). Result (d) (Theorem 5) is a significant extension of a similar result by Chernov *et al.* [15]. Yet, our proof techniques and interpretation are completely different from those in [15]. Result (e), Example 3, is a direct consequence of (a). Result (f) (Theorem 2) is completely new, but the proof is partly based on ideas which already appear in [12] in a log-loss/MDL context, and (g) is a consequence of (f). Finally, Corollary 3 can be seen as analogous to the results of Lee *et al.* [25], who showed the role of convexity of $\mathcal{F}$ for fast rates in the regression setting with squared loss.

**Acknowledgments**

This work was supported by the ARC and by NICTA, funded by the Australian Government. It was also supported in part by the IST Programme of the European Community, under the PASCAL Network of Excellence, IST-2002-506778, and by NWO Rubicon grant 680-50-1112.

# References

[1] O. Bousquet, S. Boucheron, and G. Lugosi. Introduction to statistical learning theory. In O. Bousquet, U. von Luxburg, and G. Rätsch, editors, *Advanced Lectures on Machine Learning*, volume 3176 of *Lecture Notes in Computer Science*, pages 169–207. Springer Berlin / Heidelberg, 2004.

[2] N. Cesa-Bianchi and G. Lugosi. *Prediction, learning, and games*. Cambridge University Press, 2006.

[3] O. Dekel and Y. Singer. Data-driven online to batch conversions. In Y. Weiss, B. Schölkopf, and J. Platt, editors, *Advances in Neural Information Processing Systems 18 (NIPS)*, pages 267–274, Cambridge, MA, 2006. MIT Press.

[4] J. Abernethy, A. Agarwal, P. L. Bartlett, and A. Rakhlin. A stochastic view of optimal regret through minimax duality. In *Proceedings of the 22nd Conference on Learning Theory (COLT)*, 2009.

[5] Y. Kalnishkan and M. V. Vyugin. The weak aggregating algorithm and weak mixability. *Journal of Computer and System Sciences*, 74:1228–1244, 2008.

[6] V. Vovk. A game of prediction with expert advice. In *Proceedings of the 8th Conference on Learning Theory (COLT)*, pages 51–60. ACM, 1995.

[7] E. Mammen and A. B. Tsybakov. Smooth discrimination analysis. *The Annals of Statistics*, 27(6):1808–1829, 1999.

[8] A. B. Tsybakov. Optimal aggregation of classifiers in statistical learning. *The Annals of Statistics*, 32(1):135–166, 2004.

[9] J. L. Doob. Application of the theory of martingales. In *Le Calcul de Probabilités et ses Applications. Colloques Internationaux du Centre National de la Recherche Scientifique*, pages 23–27, Paris, 1949.

[10] A. Barron and T. Cover. Minimum complexity density estimation. *IEEE Transactions on Information Theory*, 37(4):1034–1054, 1991.

[11] T. Zhang. From $\epsilon$-entropy to KL entropy: analysis of minimum information complexity density estimation. *Annals of Statistics*, 34(5):2180–2210, 2006.

[12] J. Li. *Estimation of Mixture Models*. PhD thesis, Yale University, 1999.

[13] B. Kleijn and A. van der Vaart. Misspecification in infinite-dimensional Bayesian statistics. *Annals of Statistics*, 34(2), 2006.

[14] P. Grünwald. Safe learning: bridging the gap between Bayes, MDL and statistical learning theory via empirical convexity. In *Proceedings of the 24th Conference on Learning Theory (COLT)*, 2011.

[15] A. Chernov, Y. Kalnishkan, F. Zhdanov, and V. Vovk. Supermartingales in prediction with expert advice. *Theoretical Computer Science*, 411:2647–2669, 2010.

[16] J.-Y. Audibert. Fast learning rates in statistical inference through aggregation. *Annals of Statistics*, 37(4):1591–1646, 2009.

[17] E. Vernet, R. C. Williamson, and M. D. Reid. Composite multiclass losses. In *Advances in Neural Information Processing Systems 24 (NIPS)*, 2011.

[18] P. Grünwald. *The Minimum Description Length Principle*. MIT Press, Cambridge, MA, 2007.

[19] T. van Erven, M. Reid, and R. Williamson. Mixability is Bayes risk curvature relative to log loss. In *Proceedings of the 24th Conference on Learning Theory (COLT)*, 2011.

[20] S. Arlot and P. L. Bartlett. Margin-adaptive model selection in statistical learning. *Bernoulli*, 17(2):687–713, 2011.

[21] T. Zhang. Information theoretical upper and lower bounds for statistical estimation. *IEEE Transactions on Information Theory*, 52(4):1307–1321, 2006.

[22] V. Vapnik. *Statistical Learning Theory*. Wiley, New York, 1998.

[23] J.-Y. Audibert. *PAC-Bayesian statistical learning theory*. PhD thesis, Université Paris VI, 2004.

[24] O. Catoni. *PAC-Bayesian Supervised Classification*. Lecture Notes-Monograph Series. IMS, 2007.

[25] W. Lee, P. Bartlett, and R. Williamson. The importance of convexity in learning with squared loss. *IEEE Transactions on Information Theory*, 44(5):1974–1980, 1998. Correction, Volume 54(9), 4395 (2008).

[26] A. N. Shiryaev. *Probability*. Springer-Verlag, 1996.

[27] J.-Y. Audibert. A better variance control for PAC-Bayesian classification. Preprint 905, Laboratoire de Probabilités et Modèles Aléatoires, Universités Paris 6 and Paris 7, 2004.

